# Gaussian Processes for Bayesian Classification via Hybrid Monte Carlo

**David Barber and Christopher K. I. Williams**
Neural Computing Research Group
Department of Computer Science and Applied Mathematics
Aston University, Birmingham B4 7ET, UK
d.barber@aston.ac.uk        c.k.i.williams@aston.ac.uk

## Abstract

The full Bayesian method for applying neural networks to a prediction problem is to set up the prior/hyperprior structure for the net and then perform the necessary integrals. However, these integrals are not tractable analytically, and Markov Chain Monte Carlo (MCMC) methods are slow, especially if the parameter space is high-dimensional. Using Gaussian processes we can approximate the weight space integral analytically, so that only a small number of hyperparameters need be integrated over by MCMC methods. We have applied this idea to classification problems, obtaining excellent results on the real-world problems investigated so far.

## 1  INTRODUCTION

To make predictions based on a set of training data, fundamentally we need to combine our prior beliefs about possible predictive functions with the data at hand. In the Bayesian approach to neural networks a prior on the weights in the net induces a prior distribution over functions. This leads naturally to the idea of specifying our beliefs about functions more directly. Gaussian Processes (GPs) achieve just that, being examples of stochastic process priors over functions that allow the efficient computation of predictions. It is also possible to show that a large class of neural network models converge to GPs in the limit of an infinite number of hidden units (Neal, 1996). In previous work (Williams and Rasmussen, 1996) we have applied GP priors over functions to the problem of predicting a real-valued output, and found that the method has comparable performance to other state-of-the-art methods. This paper extends the use of GP priors to classification problems.

The GPs we use have a number of adjustable hyperparameters that specify quantities like the length scale over which smoothing should take place. Rather than

optimizing these parameters (e.g. by maximum likelihood or cross-validation methods) we place priors over them and use a Markov Chain Monte Carlo (MCMC) method to obtain a sample from the posterior which is then used for making predictions. An important advantage of using GPs rather than neural networks arises from the fact that the GPs are characterized by a few (say ten or twenty) hyperparameters, while the networks have a similar number of hyperparameters but many (e.g. hundreds) of weights as well, so that MCMC integrations for the networks are much more difficult.

We first briefly review the regression framework as our strategy will be to transform the classification problem into a corresponding regression problem by dealing with the input values to the logistic transfer function. In section 2.1 we show how to use Gaussian processes for classification when the hyperparameters are fixed, and then describe the integration over hyperparameters in section 2.3. Results of our method as applied to some well known classification problems are given in section 3, followed by a brief discussion and directions for future research.

### 1.1 Gaussian Processes for regression

We outline the GP method as applied to the prediction of a real valued output $y_* = y(x_*)$ for a new input value $x_*$, given a set of training data $\mathcal{D} = \{(x_i, t_i),\ i = 1 \dots n\}$

Given a set of inputs $x_*, x_1, \dots x_n$, a GP allows us to specify how correlated we expect their corresponding outputs $y = (y(x_1), y(x_2), \dots, y(x_n))$ to be. We denote this prior over functions as $P(y)$, and similarly, $P(y_*, y)$ for the joint distribution including $y_*$. If we also specify $P(t|y)$, the probability of observing the particular values $t = (t_1, \dots t_n)^T$ given actual values $y$ (i.e. a noise model) then

$$P(y_*|t) = \int P(y_*, y|t)dy = \frac{1}{P(t)} \int P(y_*, y)P(t|y)dy \qquad (1)$$

Hence the predictive distribution for $y_*$ is found from the marginalization of the product of the prior and the noise model. If $P(t|y)$ and $P(y_*, y)$ are Gaussian then $P(y_*|t)$ is a Gaussian whose mean and variance can be calculated using matrix computations involving matrices of size $n \times n$. Specifying $P(y_*, y)$ to be a multidimensional Gaussian (for all values of $n$ and placements of the points $x_*, x_1, \dots x_n$) means that the prior over functions is a GP. More formally, a stochastic process is a collection of random variables $\{Y(x)|x \in X\}$ indexed by a set $X$. In our case $X$ will be the input space with dimension $d$, the number of inputs. A GP is a stochastic process which can be fully specified by its mean function $\mu(x) = E[Y(x)]$ and its covariance function $C(x, x') = E[(Y(x) - \mu(x))(Y(x') - \mu(x'))]$; any finite set of $Y$-variables will have a joint multivariate Gaussian distribution. Below we consider GPs which have $\mu(x) \equiv 0$.

## 2  GAUSSIAN PROCESSES FOR CLASSIFICATION

For simplicity of exposition, we will present our method as applied to two class problems as the extension to multiple classes is straightforward.

By using the logistic transfer function $\sigma$ to produce an output which can be interpreted as $\pi(x)$, the probability of the input $x$ belonging to class 1, the job of specifying a prior over functions $\pi$ can be transformed into that of specifying a prior over the input to the transfer function. We call the input to the transfer function the *activation*, and denote it by $y$, with $\pi(x) = \sigma(y(x))$. For input $x_i$, we will denote the corresponding probability and activation by $\pi_i$ and $y_i$ respectively.

To make predictions when using fixed hyperparameters we would like to compute $\hat{\pi}_* = \int \pi_* P(\pi_*|t) \, d\pi_*$, which requires us to find $P(\pi_*|t) = P(\pi(x_*)|t)$ for a new input $x_*$. This can be done by finding the distribution $P(y_*|t)$ ($y_*$ is the activation of $\pi_*$) and then using the appropriate Jacobian to transform the distribution. Formally the equations for obtaining $P(y_*|t)$ are identical to equation 1. However, even if we use a GP prior so that $P(y_*, y)$ is Gaussian, the usual expression for $P(t|y) = \prod_i \pi_i^{t_i}(1 - \pi_i)^{1-t_i}$ for classification data (where the $t$'s take on values of 0 or 1), means that the marginalization to obtain $P(y_*|t)$ is no longer analytically tractable.

We will employ Laplace's approximation, i.e. we shall approximate the integrand $P(y_*, y|t, \theta)$ by a Gaussian distribution centred at a maximum of this function with respect to $y_*, y$ with an inverse covariance matrix given by $-\nabla\nabla \log P(y_*, y|t, \theta)$. The necessary integrations (marginalization) can then be carried out analytically (see, e.g. Green and Silverman (1994) §5.3) and we provide a derivation in the following section.

## 2.1   Maximizing $P(y_*, y|t)$

Let $y_+$ denote $(y_*, y)$, the complete set of activations. By Bayes' theorem $\log P(y_+|t) = \log P(t|y) + \log P(y_+) - \log P(t)$, and let $\Psi_+ = \log P(t|y) + \log P(y_+)$. As $P(t)$ does not depend on $y_+$ (it is just a normalizing factor), the maximum of $P(y_+|t)$ is found by maximizing $\Psi_+$ with respect to $y_+$. We define $\Psi$ similarly in relation to $P(y|t)$. Using $\log P(t_i|y_i) = t_i y_i - \log(1 + e^{y_i})$, we obtain

$$\Psi_+ = t^T y - \sum_{i=1}^{n} \log(1 + e^{y_i}) - \frac{1}{2}y_+^T K_+^{-1} y_+ - \frac{1}{2}\log|K_+| - \frac{n+1}{2}\log 2\pi \quad (2)$$

$$\Psi = t^T y - \sum_{i=1}^{n} \log(1 + e^{y_i}) - \frac{1}{2}y^T K^{-1} y - \frac{1}{2}\log|K| - \frac{n}{2}\log 2\pi \quad (3)$$

where $K_+$ is the covariance matrix of the GP evaluated at $x_1, \ldots x_n, x_*$. $K_+$ can be partitioned in terms of an $n \times n$ matrix $K$, a $n \times 1$ vector $k$ and a scalar $k_*$, viz.

$$K_+ = \begin{pmatrix} K & k \\ k^T & k_* \end{pmatrix} \quad (4)$$

As $y_*$ only enters into equation 2 in the quadratic prior term and has no data point associated with it, maximizing $\Psi_+$ with respect to $y_+$ can be achieved by first maximizing $\Psi$ with respect to $y$ and then doing the further quadratic optimization to determine the posterior mean $\hat{y}_*$. To find a maximum of $\Psi$ we use the Newton-Raphson (or Fisher scoring) iteration $y^{new} = y - (\nabla\nabla\Psi)^{-1}\nabla\Psi$. Differentiating equation 3 with respect to $y$ we find

$$\nabla\Psi = (t - \pi) - K^{-1}y \quad (5)$$

$$\nabla\nabla\Psi = -K^{-1} - W \quad (6)$$

where $W = diag(\pi_1(1 - \pi_1), .., \pi_n(1 - \pi_n))$, which gives the iterative equation[1],

$$y^{new} = (K^{-1} + W)^{-1}W(y + W^{-1}(t - \pi)) \quad (7)$$

Given a converged solution $\tilde{y}$ for $y$, $\hat{y}_*$ can easily be found using $y_* = k^T K^{-1} \tilde{y} = k^T(t - \tilde{\pi})$. $var(y_*)$ is given by $(K_+^{-1} + W_+)_{(n+1)(n+1)}^{-1}$, where $W_+$ is the $W$ matrix with a zero appended in the $(n+1)$th diagonal position.

Given the (Gaussian) distribution of $y_*$ we then wish to find the mean of the distribution of $P(\pi_*|t)$ which is found from $\hat{\pi}_* = \int \sigma(y_*)P(y_*|t)$. We calculate this by approximating the sigmoid by a set of five cumulative normal densities (erf) that interpolate the sigmoid at chosen points. This leads to a very fast and accurate analytic approximation for the mean class prediction.

The justification of Laplace's approximation in our case is somewhat different from the argument usually put forward, e.g. for asymptotic normality of the maximum likelihood estimator for a model with a finite number of parameters. This is because the dimension of the problem grows with the number of data points. However, if we consider the "infill asymptotics", where the number of data points in a *bounded* region increases, then a local average of the training data at any point $x$ will provide a tightly localized estimate for $\pi(x)$ and hence $y(x)$, so we would expect the distribution $P(y)$ to become more Gaussian with increasing data.

## 2.2 Parameterizing the covariance function

There are many reasonable choices for the covariance function. Formally, we are required to specify functions which will generate a non-negative definite covariance matrix for any set of points $(x_1, \ldots, x_k)$. From a modelling point of view we wish to specify covariances so that points with nearby inputs will give rise to similar predictions. We find that the following covariance function works well:

$$C(x, x') = v_0 \exp\left\{ -\frac{1}{2} \sum_{l=1}^{d} w_l(x_l - x_l')^2 \right\} \tag{8}$$

where $x_l$ is the $l$th component of $x$ and $\theta = \log(v_0, w_1, \ldots, w_d)$ plays the role of hyperparameters[2].

We define the hyperparameters to be the log of the variables in equation 8 since these are positive scale-parameters. This covariance function has been studied by Sacks *et al* (1989) and can be obtained from a network of Gaussian radial basis functions in the limit of an infinite number of hidden units (Williams, 1996).

The $w_l$ parameters in equation 8 allow a different length scale on each input dimension. For irrelevant inputs, the corresponding $w_l$ will become small, and the model will ignore that input. This is closely related to the Automatic Relevance Determination (ARD) idea of MacKay and Neal (Neal, 1996). The $v_0$ variable gives the overall scale of the prior; in the classification case, this specifies if the $\pi$ values will typically be pushed to 0 or 1, or will hover around 0.5.

## 2.3 Integration over the hyperparameters

Given that the GP contains adjustable hyperparameters, how should they be adapted given the data? Maximum likelihood or (generalized) cross-validation methods are often used, but we will prefer a Bayesian solution. A prior distribution over the hyperparameters $P(\theta)$ is modified using the training data to obtain the posterior distribution $P(\theta|t) \propto P(t|\theta)P(\theta)$. To make predictions we integrate

the predicted probabilities over the posterior; for example, the mean value $\overline{\pi}(\boldsymbol{x}_*)$ for test input $\boldsymbol{x}_*$ is given by

$$\overline{\pi}(\boldsymbol{x}_*) = \int \hat{\pi}(\boldsymbol{x}_*|\boldsymbol{\theta})P(\boldsymbol{\theta}|\boldsymbol{t})d\boldsymbol{\theta}, \tag{9}$$

where $\hat{\pi}(\boldsymbol{x}_*|\boldsymbol{\theta})$ is the mean prediction for a fixed value of the hyperparameters, as given in section 2.

For the regression problem $P(\boldsymbol{t}|\boldsymbol{\theta})$ can be calculated exactly using $P(\boldsymbol{t}|\boldsymbol{\theta}) = \int P(\boldsymbol{t}|\boldsymbol{y})P(\boldsymbol{y}|\boldsymbol{\theta})d\boldsymbol{y}$, but this integral is not analytically tractable for the classification problem. Again we use Laplace's approximation and obtain[3]

$$\log P(\boldsymbol{t}|\boldsymbol{\theta}) \simeq \Psi(\tilde{\boldsymbol{y}}) + \frac{1}{2}|K^{-1} + W| + \frac{n}{2}\log 2\pi \tag{10}$$

where $\tilde{\boldsymbol{y}}$ is the converged iterate of equation 7. We denote the right-hand side of equation 10 by $\log P_a(\boldsymbol{t}|\boldsymbol{\theta})$ (where $a$ stands for approximate).

The integration over $\boldsymbol{\theta}$-space also cannot be done analytically, and we employ a Markov Chain Monte Carlo method. We have used the Hybrid Monte Carlo (HMC) method of Duane et al (1987), with broad Gaussian hyperpriors on the parameters.

HMC works by creating a fictitious dynamical system in which the hyperparameters are regarded as position variables, and augmenting these with momentum variables $\boldsymbol{p}$. The purpose of the dynamical system is to give the hyperparameters "inertia" so that random-walk behaviour in $\boldsymbol{\theta}$-space can be avoided. The total energy, $\mathcal{H}$, of the system is the sum of the kinetic energy, $\mathcal{K} = \boldsymbol{p}^T\boldsymbol{p}/2$ and the potential energy, $\mathcal{E}$. The potential energy is defined such that $p(\boldsymbol{\theta}|D) \propto \exp(-\mathcal{E})$, i.e. $\mathcal{E} = -\log P(\boldsymbol{t}|\boldsymbol{\theta}) - \log P(\boldsymbol{\theta})$. In practice $\log P_a(\boldsymbol{t}|\boldsymbol{\theta})$ is used instead of $\log P(\boldsymbol{t}|\boldsymbol{\theta})$. We sample from the joint distribution for $\boldsymbol{\theta}$ and $\boldsymbol{p}$ given by $P(\boldsymbol{\theta}, \boldsymbol{p}) \propto \exp(-\mathcal{E} - \mathcal{K})$; the marginal of this distribution for $\boldsymbol{\theta}$ is the required posterior. A sample of hyperparameters from the posterior can therefore be obtained by simply ignoring the momenta.

Sampling from the joint distribution is achieved by two steps: (i) finding new points in phase space with near-identical energies $\mathcal{H}$ by simulating the dynamical system using a discretised approximation to Hamiltonian dynamics, and (ii) changing the energy $\mathcal{H}$ by Gibbs sampling the momentum variables.

Hamilton's first order differential equations for $\mathcal{H}$ are approximated using the leapfrog method which requires the derivatives of $\mathcal{E}$ with respect to $\boldsymbol{\theta}$. Given a Gaussian prior on $\boldsymbol{\theta}$, $\log P(\boldsymbol{\theta})$ is straightforward to differentiate. The derivative of $\log P_a(\boldsymbol{\theta})$ is also straightforward, although implicit dependencies of $\tilde{\boldsymbol{y}}$ (and hence $\tilde{\pi}$) on $\boldsymbol{\theta}$ must be taken into account by using equation 5 at the maximum point to obtain $\partial\tilde{\boldsymbol{y}}/\partial\theta = (I + KW)^{-1}(\partial K/\partial\theta)(\boldsymbol{t} - \boldsymbol{\pi})$. The computation of the energy can be quite expensive as for each new $\boldsymbol{\theta}$, we need to perform the maximization required for Laplace's approximation, equation 10. The Newton-Raphson iteration was initialized each time with $\boldsymbol{\pi} = 0.5$, and continued until the mean relative difference of the elements of $W$ between consecutive iterations was less than $10^{-4}$.

The same step size $\varepsilon$ is used for all hyperparameters, and should be as large as possible while keeping the rejection rate low. We have used a trajectory made up of $L = 20$ leapfrog steps, which gave a low correlation between successive states[4]. This proposed state is then accepted or rejected using the Metropolis rule depending on

the final energy $\mathcal{H}^*$ (which is not necessarily equal to the initial energy $\mathcal{H}$ because of the discretization of Hamilton's equations).

The priors over hyperparameters were set to be Gaussian with a mean of $-3$ and a standard deviation of 3. In all our simulations a step size $\varepsilon = 0.1$ produced a very low rejection rate ($< 5\%$). The hyperparameters corresponding to the $w_l$'s were initialized to $-2$ and that for $v_0$ to 0. The sampling procedure was run for 200 iterations, and the first third of the run was discarded; this "burn-in" is intended to give the hyperparameters time to come close to their equilibrium distribution.

# 3 RESULTS

We have tested our method on two well known two-class classification problems, the Leptograpsus crabs and Pima Indian diabetes datasets and the multiclass Forensic Glass dataset[5]. We first rescale the inputs so that they have mean zero and unit variance on the training set. Our Matlab implementations for the HMC simulations for both tasks each take several hours on a SGI Challenge machine (R10000), although good results can be obtained in less time. We also tried a standard Metropolis MCMC algorithm for the Crabs problem, and found similar results, although the sampling by this method is slower than that for HMC. Comparisons with other methods are taken from Ripley (1994) and Ripley (1996).

Our results for the two-class problems are presented in Table 1: In the Leptograpsus crabs problem we attempt to classify the sex of crabs on the basis of five anatomical attributes. There are 100 examples available for crabs of each sex, making a total of 200 labelled examples. These are split into a training set of 40 crabs of each sex, making 80 training examples, with the other 120 examples used as the test set. The performance of the GP is equal to the best of the other methods reported in Ripley (1994), namely a 2 hidden unit neural network with direct input to output connections and a logistic output unit which was trained with maximum likelihood (Network(1) in Table 1).

For the Pima Indians diabetes problem we have used the data as made available by Prof. Ripley, with his training/test split of 200 and 332 examples respectively (Ripley, 1996). The baseline error obtained by simply classifying each record as coming from a diabetic gives rise to an error of 33%. Again, the GP method is comparable with the best alternative performance, with an error of around 20%.

| Table 1 | Pima | Crabs |
|---|---|---|
| Neural Network(1) | - | 3 |
| Neural Network(2) | - | 3 |
| Neural Network(3) | 75+ | - |
| Linear Discriminant | 67 | 8 |
| Logistic regression | 66 | 4 |
| MARS (degree = 1) | 75 | 4 |
| PP (4 ridge functions) | 75 | 6 |
| 2 Gaussian Mixture | 64 | - |
| Gaussian Process (HMC) | 68 | 3 |
| Gaussian Process (MAP) | 69 | 3 |

| Table 2 | Forensic Glass |
|---|---|
| Neural Network (4HU) | 23.8% |
| Linear Discriminant | 36% |
| MARS (degree = 1) | 32.2% |
| PP (5 ridge functions) | 35% |
| Gaussian Mixture | 30.8% |
| Decision Tree | 32.2% |
| Gaussian Process (MAP) | 23.3% |

Table 1: Number of test errors for the Pima Indian diabetes and Leptograpsus crabs tasks. Network(2) used two hidden units and the predictive approach (Ripley, 1994), which uses Laplace's approximation to weight each network local minimum. Network(3) had one hidden unit and was trained with maximum likelihood; the results were worse for nets with two or more hidden units (Ripley, 1996). Table 2: Percentage classification error on the Forensic Glass task.

Our method is readily extendable to multiple class problems by using the softmax function. The details of this work which will be presented elsewhere, and we simply report here our initial findings on the Forensic Glass problem (Table 2). This is a 6 class problem, consisting of 214 examples containing 9 attributes. The performance is estimated using 10 fold cross validation. Computing the MAP estimate took $\approx$ 24 hours and gave a classification error of 23.3%, comparable with the best of the other presented methods.

## 4 DISCUSSION

We have extended the work of Williams and Rasmussen (1996) to classification problems, and have demonstrated that it performs well on the datasets we have tried so far. One of the main advantages of this approach is that the number of parameters used in specifying the covariance function is typically much smaller than the number of weights and hyperparameters that are used in a neural network, and this greatly facilitates the implementation of Monte Carlo methods. Furthermore, because the Gaussian Process is a prior on function space (albeit in the activation function space), we are able to interpret our prior more readily than for a model in which the priors are on the parametrization of the function space, as in neural network models. Some of the elegance that is present using Gaussian Processes for regression is lost due to the inability to perform the required marginalisation exactly. Nevertheless, our simulation results suggest that Laplace's approximation is accurate enough to yield good results in practice. As methods based on GPs require the inversion of $n \times n$ matrices, where $n$ is the number of training examples, we are looking into methods such as query selection for large dataset problems. Other future research directions include the investigation of different covariance functions and improvements on the approximations employed.

We hope to make our MATLAB code available from http://www.ncrg.aston.ac.uk/

**Acknowledgements**

We thank Prof. B. Ripley for making available the Leptograpsus crabs and Pima Indian diabetes datasets. This work was partially supported by EPSRC grant GR/J75425, "Novel Developments in Learning Theory for Neural Networks".

## Footnotes

[1]The complexity of calculating each iteration using standard matrix methods is O($n^3$). In our implementation, however, we use conjugate gradient methods to avoid explicitly inverting matrices. In addition, by using the previous iterate $y$ as an initial guess for the conjugate gradient solution to equation 7, the iterates are computed an order of magnitude faster than using standard algorithms.

[2] We call $\theta$ the hyperparameters rather than parameters as they correspond closely to hyperparameters in neural networks.

[3]This requires $O(n^3)$ computation.

[4]In our experiments, where $\theta$ is only 7 or 8 dimensional, we found the trajectory length needed is much shorter than that for neural network HMC implementations.

[5] All available from `http://markov.stats.ox.ac.uk/pub/PRNN`.

## References

Duane, S., A. D. Kennedy, B. J. Pendleton, and D. Roweth (1987). Hybrid Monte Carlo. *Physics Letters B* **195**, 216–222.

Green, P. J.and Silverman, B. W. (1994). *Nonparametric regression and generalized linear models*. Chapman and Hall.

Neal, R. M. (1996). *Bayesian Learning for Neural Networks*. Springer. Lecture Notes in Statistics 118.

Ripley, B. (1996). *Pattern Recognition and Neural Networks*. Cambridge.

Ripley, B. D. (1994). Flexible Non-linear Approaches to Classification. In V. Cherkassy, J. H. Friedman, and H. Wechsler (Eds.), *From Statistics to Neural Networks*, pp. 105–126. Springer.

Sacks, J., W. J. Welch, T. J. Mitchell, and H. P. Wynn (1989). Design and analysis of computer experiments. *Statistical Science* **4**(4), 409–435.

Williams, C. K. I. Computing with infinite networks. This volume.

Williams, C. K. I. and C. E. Rasmussen (1996). Gaussian processes for regression. In D. S. Touretzky, M. C. Mozer, and M. E. Hasselmo (Eds.), *Advances in Neural Information Processing Systems 8*, pp. 514–520. MIT Press.